# Compact EEPROM-based Weight Functions

**A. Kramer, C. K. Sin, R. Chu, and P. K. Ko**
Department of Electrical Engineering and Computer Science
University of California at Berkeley
Berkeley, CA 94720

## Abstract

We are focusing on the development of a highly compact neural net weight function based on the use of EEPROM devices. These devices have already proven useful for analog weight storage, but existing designs rely on the use of conventional voltage multiplication as the weight function, requiring additional transistors per synapse. A parasitic capacitance between the floating gate and the drain of the EEPROM structure leads to an unusual I-V characteristic which can be used to advantage in designing a compact synapse. This novel behavior is well characterized by a model we have developed. A single-device circuit results in a 1-quadrant synapse function which is nonlinear, though monotonic. A simple extension employing 2 EEPROMs results in a 2 quadrant function which is much more linear. This approach offers the potential for more than a ten-fold increase in the density of neural net implementations.

## 1 INTRODUCTION - ANALOG WEIGHTING

The recent surge of interest in neural networks and parallel analog computation has motivated the need for compact analog computing blocks. Analog weighting is an important computational function of this class. Analog weighting is the combining of two analog values, one of which is typically varying (the input) and one of which is typically fixed (the weight) or at least varying more slowly. The varying value is "weighted" by the fixed value through the "weighting function", typically multiplication. Analog weighting is most interesting when the overall computational task involves computing the "weighted sum of the inputs." That is, to compute $\sum_{i=1}^{n} f(w_i, v_i)$ where $f()$ is the weighting function and $W = \{w_1, w_2, ..., w_n\}$ and

$V = \{v_1, v_2, ..., v_n\}$ are the n-dimensional analog-valued weight and input vectors. This weighted sum is simply the dot product in the case where the weighting function is multiplication.

For large n, the only way to perform this computation efficiently is to use compact weighting functions and to take advantage of current summing. Using "conductive multiplication" as the weighting function (weights stored as conductances of single devices) results in an efficient implementation such as that shown in figure 1a. This implementation is probably optimal, but in practice it is not possible to implement small single-device programmable conductances which are linear.

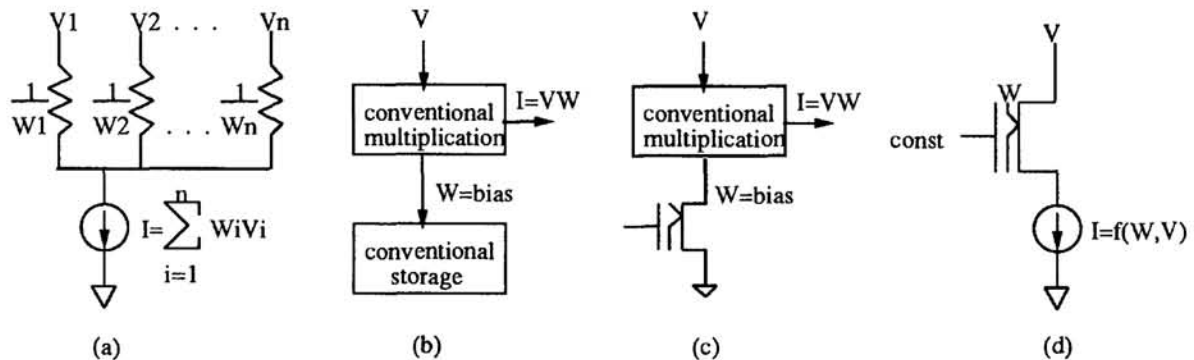

Figure 1: Weighting function implementations: (a) ideal, (b) conventional, (c) EEPROM-based storage, (d) compact EEPROM-based nonlinear weight function

## 1.1   CONVENTIONAL APPROACHES

The problem of implementing analog weighting is often divided into the separate tasks of storing the fixed value (the weight) and combining the two analog values through the weighting function (figure 1b). Conventional approaches to storing a fixed analog weight value are to use either digital storage with some form of D/A conversion or to use volatile analog storage, which requires a large capacitor. Both of these storage technologies require a large area.

The simplest and most widespread weighting function is multiplication [$f(w, i) = wi$]. Multiplication is attractive because of its mathematical and computational simplicity. Multiplication is also a fairly straightforward operation to implement in analog circuitry. When conventional technologies are used for weight storage, the additional area required to provide a multiplication function is not significant. Of course, the problem with this approach is that since a large area is required for weight storage, the result is not sufficiently compact.

## 2   EEPROMS

EEPROMs are "electrically erasable, programmable, read-only memories". They are essentially a JFET with a floating gate and a thin-oxide tunneling region between the floating gate and the drain (figure 2). A sufficiently high field across the tunneling oxide will cause electrons to tunnel into or out of the floating gate,

effectively altering the threshold voltage of the device as seen from the top gate. Normal operating (reading) voltages are sufficiently small to cause only insignificant "disturbance programming" of the charge on the floating gate, so an EEPROM can be viewed as a compact storage capacitor with a very long storage lifetime.

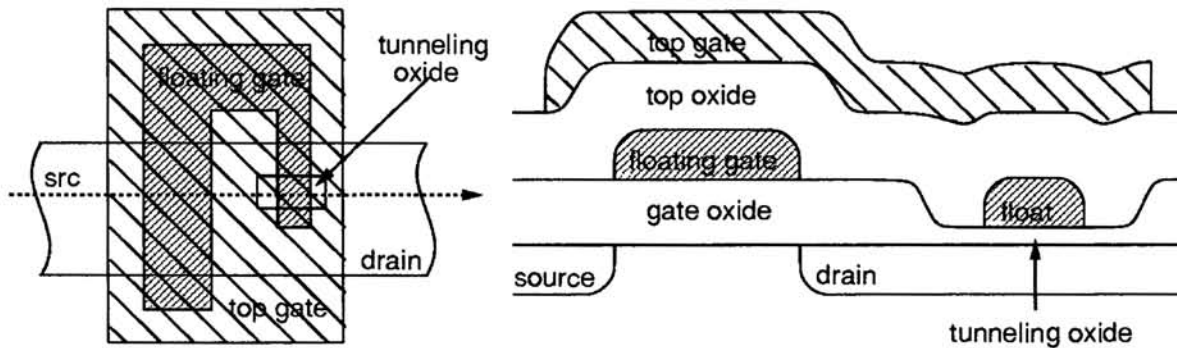

Figure 2: EEPROM layout and cross section

Several groups have found that charge leakage on EEPROMs is sufficiently small to guarantee that the threshold of a device can be retained with 4-8 bits of precision for a period of years [Kramer, 1989][Holler, 1989]. There are several drawbacks to the use of EEPROMs. Correct programming of these devices to the desired value is hard to control and requires feedback. While the programming time for a single device is less than a millisecond, because devices must be programmed one-at-a-time, the time to program all the devices on a chip can be prohibitive. In addition, fabrication of EEPROMs is a non-standard process requiring several additional masks and the ability to make a thin tunneling oxide.

## 2.1  EEPROM-BASED WEIGHT STORAGE

The most straightforward manner to use an EEPROM in a weighting function is to store the weight with the device. For example, the threshold of an EEPROM device could be programmed to produce the desired bias current for an analog amplifier (figure 1c). There are two advantages to this approach. Firstly, the weight storage mechanism is divorced from the actual weight function computation and hence places few constraints on it, and secondly, if the EEPROM is used in a static mode (all applied voltages are constant), the exact I-V characteristics of the EEPROM device are inconsequential.

The major disadvantage of this approach is that of inefficiency, as additional circuitry is needed to perform the weight function computation. An example of this can be seen in a recent EEPROM-based neural net implementation developed by the Intel corporation [Holler, 1989]. Though the weight value in this implementation is stored on only two EEPROMs, an additional 4 transistors are needed for the multiplication function. In addition, though the circuit was designed to perform multiplication the output is not quite linear under the best of conditions and, under certain conditions, exhibits severe nonlinearity. Despite these limitations, this design demonstrates the advantage of EEPROM storage technology over conventional approaches, as it is the most dense neural network implementation to date.

## 3   EEPROM I-V CHARACTERISTICS

Since linearity is difficult to implement and not a strict requirement of the weighting function, we have investigated the possibility of using the I-V characteristics of an EEPROM as the weight function. This approach has the advantage that a single device could be used for both weight storage and weight function computation, providing a very compact implementation. It is our hope that this approach will lead to useful synapses of less than $200um^2$ in area, less than a tenth the area used by the Intel synapse.

Though an EEPROM is a JFET device, a parasitic capacitance of the structure results in an I-V characteristic which is unique. Conventional use of EEPROM devices in digital circuitry does not make use of this fact, so that this effect has not before been characterized or modeled. The floating gate of an EEPROM is controlled via capacitive coupling by the top gate. In addition, the thin-ox tunneling region between the floating gate and the drain creates a parasitic capacitor between these two nodes. Though the area of this drain capacitor is small relative to that of the top-gate floating-gate overlap area, the tunneling oxide is much thinner than the insulating oxide between the two gates, resulting in a significant drain capacitance (figure 3).

We have developed a model for an EEPROM which includes this parasitic drain capacitance (figure 3). The basic contribution of this capacitance is to couple the floating-gate voltage to the drain voltage. This is most obvious when the device is saturated; while the current through a standard JFET is to first order independent of drain voltage in this region, in the case of an EEPROM, the current has a square law dependence on the drain voltage (equation 3). While this artifact of EEPROMs makes them behave poorly as current sources, it may make them more useful as single-device weighting functions.

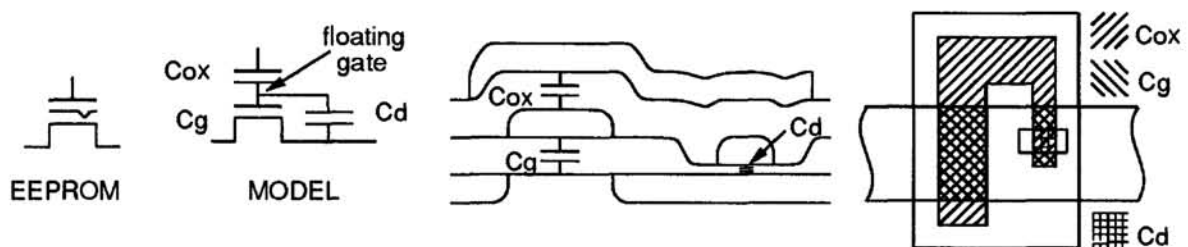

Figure 3: EEPROM model and capacitor areas

There are several ways to analyze our model depending on the level of accuracy desired [Sin, 1991]. We present here the results of simplest of these which captures the essential behavior of an EEPROM. This analysis is based on a linear channel approximation and the equations which result are similar in form to those for a normal JFET, with the addition of the dependence between the floating gate voltage and the drain voltage and all capacitive coupling factors. The equations for drain saturation voltage ($V_{ds_{sat}}$), nonsaturated drain current ($I_{ds_{lin}}$) and saturated drain current ($I_{ds_{sat}}$) are:

$$V_{ds_{sat}} = \frac{C'_g V_g - V_t(C_{ox} + C_g + C_d)}{0.5 C_{ox} + C_g} \qquad (1)$$

$$I_{ds_{lin}} = K_p \left[ \left( \frac{C_g V_g}{C_{ox} + C_g + C_d} - V_t \right) - \frac{V_{ds}^2}{2} \left( \frac{C_g - C_d}{C_{ox} + C_g + C_d} \right) \right] \qquad (2)$$

$$I_{ds_{sat}} = K_p \left[ \frac{C_g V_g + C_d V_{ds} - V_t(C_{ox} + C_g + C_d)}{0.5 C_{ox} + C_g + C_d} \right]^2 \qquad (3)$$

On EEPROM devices we have fabricated in house, our model matches measured I-V data well, especially in capturing the dependence of saturated drain current on drain voltage (figure 4).

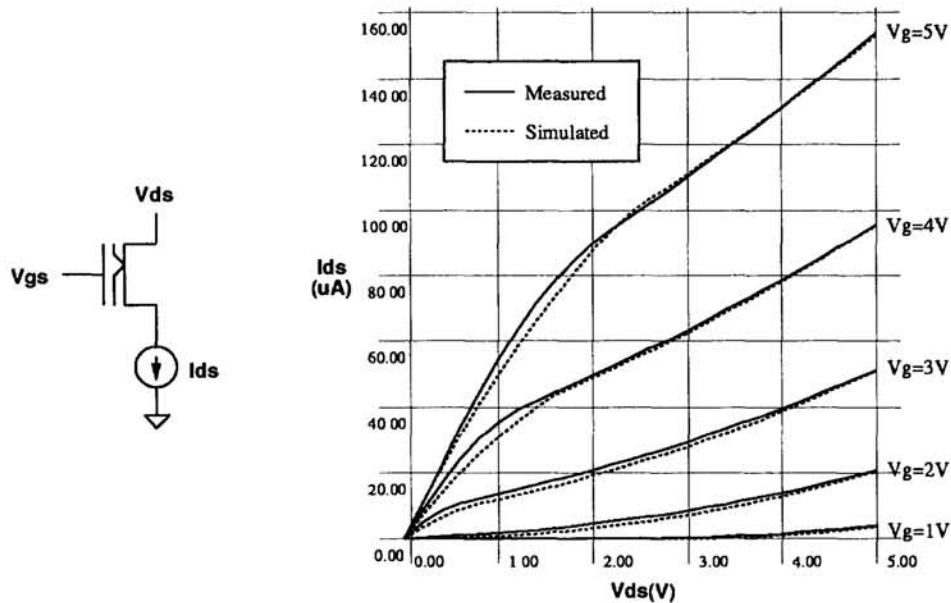

Figure 4: EEPROM I-V, measured and simulated.

## 4    EEPROM-BASED WEIGHTING FUNCTIONS

One way to make a compact weight function using an EEPROM is to use the device I-V characteristics directly. This could be accomplished by storing the weight as the device threshold voltage ($V_t$), applying the input value as the drain-source voltage ($Vds$) and setting the top gate voltage to a constant reference value (figure 1d). In this case the synapse would look exactly like the I-V measuring circuit and the weighting function would be exactly the EEPROM I-V shown in figure 4, except that rather than leaving the threshold voltage fixed and varying the gate voltage, as was done to generate the curves shown, the gate voltage would be fixed to a constant value and different curves would be generated by programming the device threshold to different values.

While extremely compact (a single device), this function is only a one quadrant function (both weight and input values must be positive or output is zero) and for

many applications this is not sufficient. An easy way to provide a two-quadrant function based on a similar approach is to use two EEPROMs configured in a common-input, differential-output ($I_{out} = I_{ds+} - I_{ds-}$) scheme, as in the circuit depicted in figure 5. By programming the EEPROMs so that one is always active and one is always inactive, the output of the weight function can now be a "positive" or a "negative" current, depending on which device is chosen. Again, the weighting function is exactly the EEPROM I-V in this case.

In addition to providing a two-quadrant function, this two-device circuit offers another interesting possibility. The same differential output scheme can be made to provide a much more linear two quadrant function if both "positive" and "negative" devices are programmed to be active (negative thresholds). The "weight" in this case is the difference in threshold values between the two devices ($W = V_{t-} - V_{t+}$). This scheme "subtracts" one device curve from the other. The model we have developed indicates that this has the effect of canceling out much of the nonlinearity and results in a function which has three distinct regions, two of which are linear in the input voltage and the weight value.

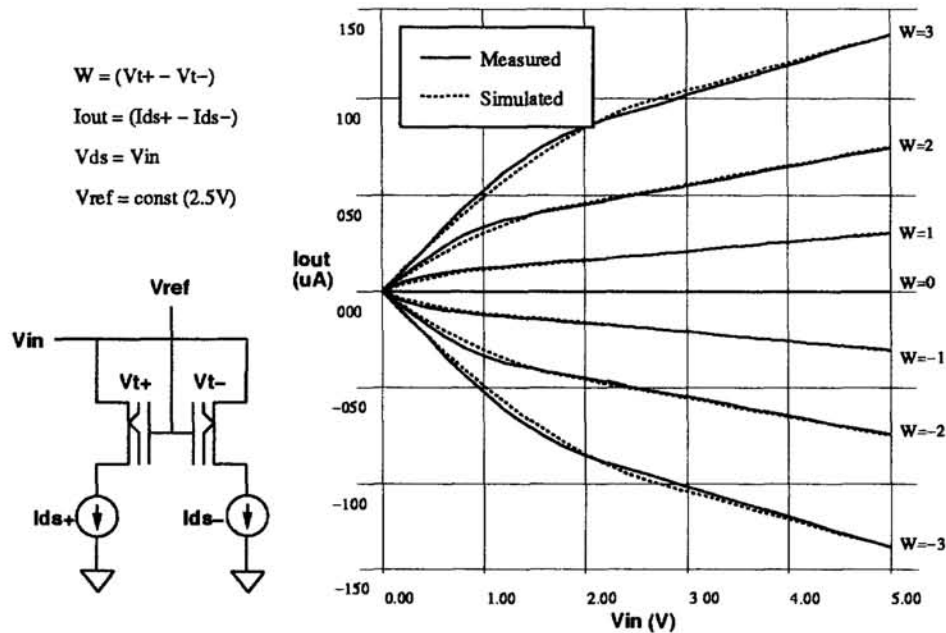

Figure 5: 2-quadrant, 2-EEPROM weighting function.

The first of these linear regions occurs when both devices are active and neither is saturated (both devices modeled by equation 2). In this case, subtracting $I_{ds-}$ from $I_{ds+}$ cancels all nonlinearities and the differential is exactly the product of the input value ($V_{ds}$) and the weight ($V_{t-} - V_{t+}$), with a scaling factor of Kp:

$$I_{ds+} - I_{ds-} = K_p V_{ds} \left( V_{t-} - V_{t+} \right) \tag{4}$$

The other linear region occurs when both devices are saturated (both modeled by equation 3). All nonlinearities also cancel in this case, but there is an offset remaining and the scaling factor is modified:

$$I_{ds+} - I_{ds-} = K_p \left( \frac{C_d}{0.5C_{ox} + C_g + C_d} \right) V_{ds} \left( V_{t-} - V_{t+} \right) +$$

$$K_p \left( V_{t-} - V_{t+} \right) \left( \frac{C_g V_g}{0.5C_{ox} + C_g + C_d} - \left( V_{t+} + V_{t-} \right) \right) \quad (5)$$

We have fabricated structures of this type and measured, as well as simulated their function characteristics. Measured data again agreed with our model (figure 5). Note that the slope in this last region [scaling factor of $K_p C_g/(0.5C_{ox} + C_g + C_d)$] will be strictly less that in the first region [scaling factor $K_p$]. The model indicates that one way to minimize this difference in slopes is to increase the size of the parasitic drain capacitance ($C_d$) relative to the gate capacitance ($C_g$).

## 5  CONCLUSIONS

While EEPROM devices have already proven useful for nonvolatile analog storage, we have discovered and characterized novel functional characteristics of the EEPROM device which should make them useful as analog weighting functions. A parasitic drain-floating gate capacitance has been included in a model which accurately captures this behavior. Several compact nonlinear EEPROM-based weight functions have been proposed, including a single-device one-quadrant function and a more linear two-device two-quadrant function. Problems such as the usability of nonlinear weighting functions, selection of optimal EEPROM device parameters and potential fanout limitations of feeding the input into a low impedance node (drain) must all be resolved before this technology can be used for a full blown implementation. Our model will be helpful in this work. The approach of using inherent device characteristics to build highly compact weighting functions promises to greatly improve the density and efficiency of massively parallel analog computation such as that performed by neural networks.

**Acknowledgements**

Research sponsored by the Air Force Office of Scientific Research (AFSOR/JSEP) under Contract Number F49620-90-C-0029.

**References**

M. Holler, et. al., (1989) "An Electrically Trainable Artificial Neural Network (ETANN) with 10240 'Floating Gate' Synapses," *Proceedings of the ICJNN-89*, Washington D. C., 1989.

A. Kramer, et. al, (1989) "EEPROM Device as a Reconfigurable Analog Element for Neural Networks," *1989 IEDM Technical Digest*, Beaver Press, Alexandria, VA, Dec. 1989.

C. K. Sin, (1990) *EEPROM as an Analog Storage Element*, Master's Thesis, Dept. of EECS, University of California at Berkeley, Berkeley, CA, Sept. 1990.